# Unlabeled data: Now it helps, now it doesn't

**Aarti Singh, Robert D. Nowak**[*]
Department of Electrical and Computer Engineering
University of Wisconsin - Madison
Madison, WI 53706
{singh@cae,nowak@engr}.wisc.edu

**Xiaojin Zhu**[†]
Department of Computer Sciences
University of Wisconsin - Madison
Madison, WI 53706
jerryzhu@cs.wisc.edu

## Abstract

Empirical evidence shows that in favorable situations *semi-supervised* learning (SSL) algorithms can capitalize on the abundance of *unlabeled* training data to improve the performance of a learning task, in the sense that fewer *labeled* training data are needed to achieve a target error bound. However, in other situations unlabeled data do not seem to help. Recent attempts at theoretically characterizing SSL gains only provide a partial and sometimes apparently conflicting explanations of whether, and to what extent, unlabeled data can help. In this paper, we attempt to bridge the gap between the practice and theory of semi-supervised learning. We develop a finite sample analysis that characterizes the value of unlabeled data and quantifies the performance improvement of SSL compared to supervised learning. We show that there are large classes of problems for which SSL can significantly outperform supervised learning, in finite sample regimes and sometimes also in terms of error convergence rates.

## 1 Introduction

Labeled data can be expensive, time-consuming and difficult to obtain in many applications. Semi-supervised learning (SSL) aims to capitalize on the abundance of unlabeled data to improve learning performance. Empirical evidence suggests that in certain favorable situations unlabeled data can help, while in other situations it does not. As a result, there have been several recent attempts [1, 2, 3, 4, 5, 6] at developing a theoretical understanding of semi-supervised learning. It is well-accepted that unlabeled data can help only if there exists a *link* between the marginal data distribution and the target function to be learnt. Two common types of links considered are the cluster assumption [7, 3, 4] which states that the target function is locally smooth over subsets of the feature space delineated by some property of the marginal density (but may not be globally smooth), and the manifold assumption [4, 6] which assumes that the target function lies on a low-dimensional manifold. Knowledge of these sets, which can be gleaned from unlabeled data, simplify the learning task. However, recent attempts at characterizing the amount of improvement possible under these links only provide a partial and sometimes apparently conflicting (for example, [4] vs. [6]) explanations of whether or not, and to what extent semi-supervised learning helps. In this paper, we bridge the gap between these seemingly conflicting views and develop a minimax framework based on finite sample bounds to identify situations in which unlabeled data help to improve learning. Our results quantify both the amount of improvement possible using SSL as well as the the relative value of unlabeled data.

We focus on learning under a cluster assumption that is formalized in the next section, and establish that there exist nonparametric classes of distributions, denoted $\mathcal{P}_{XY}$, for which the *decision sets* (over which the target function is smooth) are discernable from unlabeled data. Moreover, we show that there exist *clairvoyant* supervised learners that, given perfect knowledge of the decision sets denoted by $\mathcal{D}$, can significantly outperform any generic supervised learner $f_n$ in these

---
[*]Supported in part by the NSF grants CCF-0353079, CCF-0350213, and CNS-0519824.
[†]Supported in part by the Wisconsin Alumni Research Foundation.

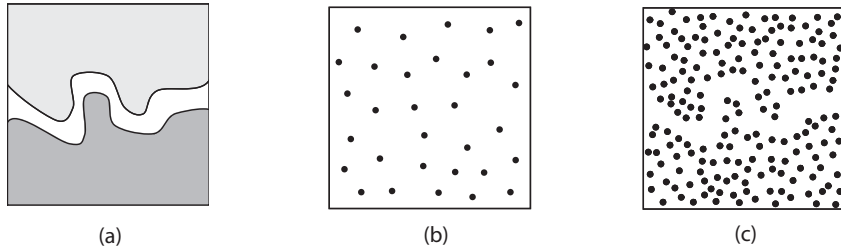

Figure 1: (a) Two separated high density sets with different labels that (b) cannot be discerned if the sample size is too small, but (c) can be estimated if sample density is high enough.

classes. That is, if $\mathcal{R}$ denotes a risk of interest, $n$ denotes the labeled data sample size, $\widehat{f}_{\mathcal{D},n}$ denotes the clairvoyant supervised learner, and $\mathbb{E}$ denotes expectation with respect to training data, then $\sup_{\mathcal{P}_{XY}} \mathbb{E}[\mathcal{R}(\widehat{f}_{\mathcal{D},n})] < \inf_{f_n} \sup_{\mathcal{P}_{XY}} \mathbb{E}[\mathcal{R}(f_n)]$. Based on this, we establish that there also exist semi-supervised learners, denoted $\widehat{f}_{m,n}$, that use $m$ unlabeled examples in addition to the $n$ labeled examples in order to estimate the decision sets, which perform as well as $\widehat{f}_{\mathcal{D},n}$, provided that $m$ grows appropriately relative to $n$. Specifically, if the error bound for $\widehat{f}_{\mathcal{D},n}$ decays polynomially (exponentially) in $n$, then the number of unlabeled data $m$ needs to grow polynomially (exponentially) with the number of labeled data $n$. We provide general results for a broad range of learning problems using finite sample error bounds. Then we examine a concrete instantiation of these general results in the regression setting by deriving minimax lower bounds on the performance of any supervised learner and compare that to upper bounds on the errors of $\widehat{f}_{\mathcal{D},n}$ and $\widehat{f}_{m,n}$.

In their seminal papers, Castelli and Cover [8, 9] suggested that in the classification setting the marginal distribution can be viewed as a mixture of class conditional distributions. If this mixture is identifiable, then the classification problem may reduce to a simple hypothesis testing problem for which the error converges exponentially fast in the number of labeled examples. The ideas in this paper are similar, except that we do not require identifiability of the mixture component densities, and show that it suffices to only approximately learn the decision sets over which the label is smooth. More recent attempts at theoretically characterizing SSL have been relatively pessimistic. Rigollet [3] establishes that for a fixed collection of distributions satisfying a cluster assumption, unlabeled data do not provide an improvement in convergence rate. A similar argument was made by Lafferty and Wasserman [4], based on the work of Bickel and Li [10], for the manifold case. However, in a recent paper, Niyogi [6] gives a constructive example of a class of distributions supported on a manifold whose complexity increases with the number of labeled examples, and he shows that the error of any supervised learner is bounded from below by a constant, whereas there exists a semi-supervised learner that can provide an error bound of $O(n^{-1/2})$, assuming infinite unlabeled data. In this paper, we bridge the gap between these seemingly conflicting views. Our arguments can be understood by the simple example shown in Fig. 1, where the distribution is supported on two component sets separated by a margin $\gamma$ and the target function is smooth over each component. Given a finite sample of data, these decision sets may or may not be discernable depending on the sampling density (see Fig. 1(b), (c)). If $\gamma$ is fixed (this is similar to fixing the class of cluster-based distributions in [3] or the manifold in [4, 10]), then given enough labeled data a supervised learner can achieve optimal performance (since, eventually, it operates in regime (c) of Fig. 1). Thus, in this example, there is no improvement due to unlabeled data in terms of the rate of error convergence for a fixed collection of distributions. However, since the true separation between the component sets is unknown, given a finite sample of data, there always exists a distribution for which these sets are indiscernible (e.g., $\gamma \to 0$). This perspective is similar in spirit to the argument in [6]. We claim that meaningful characterizations of SSL performance and quantifications of the value of unlabeled data require finite sample error bounds, and that rates of convergence and asymptotic analysis may not capture the distinctions between SSL and supervised learning. Simply stated, if the component density sets are discernable from a finite sample size $m$ of unlabeled data but not from a finite sample size $n < m$ of labeled data, then SSL can provide better performance than supervised learning. We also show that there are certain plausible situations in which SSL yields rates of convergence that cannot be achieved by any supervised learner.

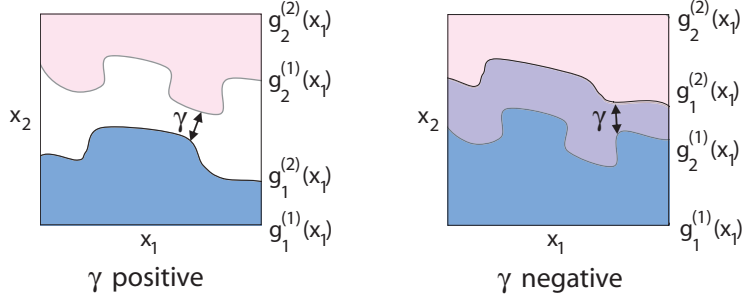

Figure 2: Margin $\gamma$ measures the minimum width of a decision set or separation between the support sets of the component marginal mixture densities. The margin is positive if the component support sets are disjoint, and negative otherwise.

## 2    Characterization of model distributions under the cluster assumption

Based on the cluster assumption [7, 3, 4], we define the following collection of joint distributions $\mathcal{P}_{XY}(\gamma) = \mathcal{P}_X \times \mathcal{P}_{Y|X}$ indexed by a margin parameter $\gamma$. Let $X, Y$ be bounded random variables with marginal distribution $P_X \in \mathcal{P}_X$ and conditional label distribution $P_{Y|X} \in \mathcal{P}_{Y|X}$, supported on the domain $\mathcal{X} = [0,1]^d$.

The marginal density $p(x) = \sum_{k=1}^{K} a_k p_k(x)$ is the mixture of a finite, but unknown, number of component densities $\{p_k\}_{k=1}^{K}$, where $K < \infty$. The unknown mixing proportions $a_k \geq a > 0$ and $\sum_{k=1}^{K} a_k = 1$. In addition, we place the following assumptions on the mixture component densities:
1. $p_k$ is supported on a unique compact, connected set $C_k \subseteq \mathcal{X}$ with Lipschitz boundaries. Specifically, we assume the following form for the component support sets: (See Fig. 2 for d=2 illustration.)

$$C_k = \{x \equiv (x_1, \dots, x_d) \in \mathcal{X} : g_k^{(1)}(x_1, \dots, x_{d-1}) \leq x_d \leq g_k^{(2)}(x_1, \dots, x_{d-1})\},$$

where $g_k^{(1)}(\cdot), g_k^{(2)}(\cdot)$ are $d-1$ dimensional Lipschitz functions with Lipschitz constant $L$.[1]
2. $p_k$ is bounded from above and below, $0 < b \leq p_k \leq B$.
3. $p_k$ is Hölder-$\alpha$ smooth on $C_k$ with Hölder constant $K_1$ [12, 13].

Let the conditional label density on $C_k$ be denoted by $p_k(Y|X = x)$. Thus, a labeled training point $(X, Y)$ is obtained as follows. With probability $a_k$, $X$ is drawn from $p_k$ and $Y$ is drawn from $p_k(Y|X = x)$. In the supervised setting, we assume access to $n$ labeled data $\mathcal{L} = \{X_i, Y_i\}_{i=1}^{n}$ drawn i.i.d according to $P_{XY} \in \mathcal{P}_{XY}(\gamma)$, and in the semi-supervised setting, we assume access to $m$ additional unlabeled data $\mathcal{U} = \{X_i\}_{i=1}^{m}$ drawn i.i.d according to $P_X \in \mathcal{P}_X$.

Let $\mathcal{D}$ denote the collection of all non-empty sets obtained as intersections of $\{C_k\}_{k=1}^{K}$ or their complements $\{C_k^c\}_{k=1}^{K}$, excluding the set $\cap_{k=1}^{K} C_k^c$ that does not lie in the support of the marginal density. Observe that $|\mathcal{D}| \leq 2^K$, and in practical situations the cardinality of $\mathcal{D}$ is much smaller as only a few of the sets are non-empty. The cluster assumption is that the target function will be smooth on each set $D \in \mathcal{D}$, hence the sets in $\mathcal{D}$ are called *decision sets*. At this point, we do not consider a specific target function.

The collection $\mathcal{P}_{XY}$ is indexed by a margin parameter $\gamma$, which denotes the minimum width of a decision set or separation between the component support sets $C_k$. The margin $\gamma$ is assigned a positive sign if there is no overlap between components, otherwise it is assigned a negative sign as illustrated in Figure 2. Formally, for $j, k \in \{1, \dots, K\}$, let

$$d_{jk} := \min_{p,q \in \{1,2\}} \|g_j^{(p)} - g_k^{(q)}\|_\infty \quad j \neq k, \qquad d_{kk} := \|g_k^{(1)} - g_k^{(2)}\|_\infty.$$

Then the margin is defined as

$$\gamma = \sigma \cdot \min_{j,k \in \{1,\dots,K\}} d_{jk}, \quad \text{where} \quad \sigma = \begin{cases} 1 & \text{if } C_j \cap C_k = \emptyset \; \forall j \neq k \\ -1 & \text{otherwise} \end{cases}.$$

## 3   Learning Decision Sets

Ideally, we would like to break a given learning task into separate subproblems on each $D \in \mathcal{D}$ since the target function is smooth on each decision set. Note that the marginal density $p$ is also smooth within each decision set, but exhibits jumps at the boundaries since the component densities are bounded away from zero. Hence, the collection $\mathcal{D}$ can be learnt from unlabeled data as follows:

1) *Marginal density estimation* — The procedure is based on the sup-norm kernel density estimator proposed in [14]. Consider a uniform square grid over the domain $\mathcal{X} = [0,1]^d$ with spacing $2h_m$, where $h_m = \kappa_0 \left( (\log m)^2/m \right)^{1/d}$ and $\kappa_0 > 0$ is a constant. For any point $x \in \mathcal{X}$, let $[x]$ denote the closest point on the grid. Let $G$ denote the kernel and $H_m = h_m \mathbf{I}$, then the estimator of $p(x)$ is

$$\widehat{p}(x) = \frac{1}{mh_m^d} \sum_{i=1}^m G(H_m^{-1}(X_i - [x])).$$

2) *Decision set estimation* — Two points $x_1, x_2 \in \mathcal{X}$ are said to be *connected*, denoted by $x_1 \leftrightarrow x_2$, if there exists a sequence of points $x_1 = z_1, z_2, \ldots, z_{l-1}, z_l = x_2$ such that $z_2, \ldots, z_{l-1} \in \mathcal{U}$, $\|z_j - z_{j+1}\| \leq 2\sqrt{d}h_m$, and for all points that satisfy $\|z_i - z_j\| \leq h_m \log m$, $|\widehat{p}(z_i) - \widehat{p}(z_j)| \leq \delta_m :=$ $(\log m)^{-1/3}$. That is, there exists a sequence of $2\sqrt{d}h_m$-dense unlabeled data points between $x_1$ and $x_2$ such that the marginal density varies smoothly along the sequence. All points that are pairwise connected specify an empirical decision set. This decision set estimation procedure is similar in spirit to the semi-supervised learning algorithm proposed in [15]. In practice, these sequences only need to be evaluated for the test and labeled training points.

The following lemma shows that if the margin is large relative to the average spacing $m^{-1/d}$ between unlabeled data points, then with high probability, two points are connected if and only if they lie in the same decision set $D \in \mathcal{D}$, provided the points are not too close to the decision boundaries. The proof sketch of the lemma and all other results are deferred to Section 7.

**Lemma 1.** *Let $\partial D$ denote the boundary of $D$ and define the set of boundary points as*

$$\mathcal{B} = \{x : \inf_{z \in \cup_{D \in \mathcal{D}} \partial D} \|x - z\| \leq 2\sqrt{d}h_m\}.$$

*If $|\gamma| > C_o(m/(\log m)^2)^{-1/d}$, where $C_o = 6\sqrt{d}\kappa_0$, then for all $p \in \mathcal{P}_X$, all pairs of points $x_1, x_2 \in supp(p) \setminus \mathcal{B}$ and all $D \in \mathcal{D}$, with probability $> 1 - 1/m$,*

$$x_1, x_2 \in D \quad \text{if and only if} \quad x_1 \leftrightarrow x_2$$

*for large enough $m \geq m_0$, where $m_0$ depends only on the fixed parameters of the class $\mathcal{P}_{XY}(\gamma)$.*

## 4   SSL Performance and the Value of Unlabeled Data

We now state our main result that characterizes the performance of SSL relative to supervised learning and follows as a corollary to the lemma stated above. Let $\mathcal{R}$ denote a risk of interest and $\mathcal{E}(\widehat{f}) = \mathcal{R}(\widehat{f}) - \mathcal{R}^*$, where $\mathcal{R}^*$ is the infimum risk over all possible learners.

**Corollary 1.** *Assume that the excess risk $\mathcal{E}$ is bounded. Suppose there exists a clairvoyant supervised learner $\widehat{f}_{\mathcal{D},n}$, with perfect knowledge of the decision sets $\mathcal{D}$, for which the following finite sample upper bound holds*

$$\sup_{\mathcal{P}_{XY}(\gamma)} \mathbb{E}[\mathcal{E}(\widehat{f}_{\mathcal{D},n})] \leq \epsilon_2(n).$$

*Then there exists a semi-supervised learner $\widehat{f}_{m,n}$ such that if $|\gamma| > C_o(m/(\log m)^2)^{-1/d}$,*

$$\sup_{\mathcal{P}_{XY}(\gamma)} \mathbb{E}[\mathcal{E}(\widehat{f}_{m,n})] \leq \epsilon_2(n) + O\left( \frac{1}{m} + n\left( \frac{m}{(\log m)^2} \right)^{-1/d} \right).$$

This result captures the essence of the relative characterization of semi-supervised and supervised learning for the margin based model distributions. It suggests that if the sets $\mathcal{D}$ are discernable using unlabeled data (the margin is large enough compared to average spacing between unlabeled data points), then there exists a semi-supervised learner that can perform as well as a supervised learner with clairvoyant knowledge of the decision sets, provided $m \gg n$ so that $(n/\epsilon_2(n))^d =$

$O(m/(\log m)^2)$ implying that the additional term in the performace bound for $\widehat{f}_{m,n}$ is negligible compared to $\epsilon_2(n)$. This indicates that if $\epsilon_2(n)$ decays polynomially (exponentially) in $n$, then $m$ needs to grow polynomially (exponentially) in $n$.

Further, suppose that the following finite sample lower bound holds for any supervised learner:

$$\inf_{f_n} \sup_{\mathcal{P}_{XY}(\gamma)} \mathbb{E}[\mathcal{E}(f_n)] \geq \epsilon_1(n).$$

If $\epsilon_2(n) < \epsilon_1(n)$, then there exists a clairvoyant supervised learner with perfect knowledge of the decision sets that outperforms any supervised learner that does not have this knowledge. Hence, Corollary 1 implies that SSL can provide better performance than any supervised learner provided (i) $m \gg n$ so that $(n/\epsilon_2(n))^d = O(m/(\log m)^2)$, and (ii) knowledge of the decision sets simplifies the supervised learning task, so that $\epsilon_2(n) < \epsilon_1(n)$. In the next section, we provide a concrete application of this result in the regression setting. As a simple example in the binary classification setting, if $p(x)$ is supported on two disjoint sets and if $P(Y = 1|X = x)$ is strictly greater than $1/2$ on one set and strictly less than $1/2$ on the other, then perfect knowledge of the decision sets reduces the problem to a hypothesis testing problem for which $\epsilon_2(n) = O(e^{-\zeta n})$, for some constant $\zeta > 0$. However, if $\gamma$ is small relative to the average spacing $n^{-1/d}$ between labeled data points, then $\epsilon_1(n) = cn^{-1/d}$ where $c > 0$ is a constant. This lower bound follows from the minimax lower bound proofs for regression in the next section. Thus, an exponential improvement is possible using semi-supervised learning provided $m$ grows exponentially in $n$.

## 5 Density-adaptive Regression

Let $Y$ denote a continuous and bounded random variable. Under squared error loss, the target function is $f(x) = \mathbb{E}[Y|X = x]$, and $\mathcal{E}(\widehat{f}) = \mathbb{E}[(\widehat{f}(X) - f(X))^2]$. Recall that $p_k(Y|X = x)$ is the conditional density on the $k$-th component and let $\mathbb{E}_k$ denote expectation with respect to the corresponding conditional distribution. The regression function on each component is $f_k(x) = \mathbb{E}_k[Y|X = x]$ and we assume that for $k = 1, \ldots, K$

1. $f_k$ is uniformly bounded, $|f_k| \leq M$.
2. $f_k$ is Hölder-$\alpha$ smooth on $C_k$ with Hölder constant $K_2$.

This implies that the overall regression function $f(x)$ is piecewise Hölder-$\alpha$ smooth; i.e., it is Hölder-$\alpha$ smooth on each $D \in \mathcal{D}$, except possibly at the component boundaries. [2] Since a Hölder-$\alpha$ smooth function can be locally well-approximated by a Taylor polynomial, we propose the following semi-supervised learner that performs local polynomial fits within each empirical decision set, that is, using training data that are connected as per the definition in Section 3. While a spatially uniform estimator suffices when the decision sets are discernable, we use the following spatially adaptive estimator proposed in Section 4.1 of [12]. This ensures that when the decision sets are indiscernible using unlabeled data, the semi-supervised learner still achieves an error bound that is, up to logarithmic factors, no worse than the minimax lower bound for supervised learners.

$$\widehat{f}_{m,n,x}(\cdot) = \arg\min_{f' \in \Gamma} \sum_{i=1}^{n} (Y_i - f'(X_i))^2 \mathbf{1}_{x \leftrightarrow X_i} + \text{pen}(f') \quad \text{and} \quad \widehat{f}_{m,n}(x) \equiv \widehat{f}_{m,n,x}(x)$$

Here $\mathbf{1}_{x \leftrightarrow X_i}$ is the indicator of $x \leftrightarrow X_i$ and $\Gamma$ denotes a collection of piecewise polynomials of degree $\lceil \alpha \rceil$ (the maximal integer $< \alpha$) defined over recursive dyadic partitions of the domain $\mathcal{X} = [0,1]^d$ with cells of sidelength between $2^{-\lceil \log(n/\log n)/(2\alpha+d) \rceil}$ and $2^{-\lceil \log(n/\log n)/d \rceil}$. The penalty term $\text{pen}(f')$ is proportional to $\log(\sum_{i=1}^{n} \mathbf{1}_{x \leftrightarrow X_i}) \# f'$, where $\# f'$ denotes the number of cells in the recursive dyadic partition on which $f'$ is defined. It is shown in [12] that this estimator yields a finite sample error bound of $n^{-2\alpha/(2\alpha+d)}$ for Hölder-$\alpha$ smooth functions, and $\max\{n^{-2\alpha/(2\alpha+d)}, n^{-1/d}\}$ for piecewise Hölder-$\alpha$ functions, ignoring logarithmic factors.

Using these results from [12] and Corollary 1, we now state finite sample upper bounds on the semi-supervised learner (SSL) described above. Also, we derive finite sample minimax lower bounds on the performance of any supervised learner (SL). Our main results are summarized in the following table, for model distributions characterized by various values of the margin parameter $\gamma$. A sketch

of the derivations of the results is provided in Section 7.3. Here we assume that dimension $d \geq 2\alpha/(2\alpha - 1)$. If $d < 2\alpha/(2\alpha - 1)$, then the supervised learning error due to to not resolving the decision sets (which behaves like $n^{-1/d}$) is smaller than error incurred in estimating the target function itself (which behaves like $n^{-2\alpha/(2\alpha+d)}$). Thus, when $d < 2\alpha/(2\alpha - 1)$, the supervised regression error is dominated by the error in smooth regions and there appears to be no benefit to using a semi-supervised learner. In the table, we suppress constants and log factors in the bounds, and also assume that $m \gg n^{2d}$ so that $(n/\epsilon_2(n))^d = O(m/(\log m)^2)$. The constants $c_o$ and $C_o$ only depend on the fixed parameters of the class $\mathcal{P}_{XY}(\gamma)$ and do not depend on $\gamma$.

| Margin range $\gamma$ | SSL upper bound $\epsilon_2(n)$ | SL lower bound $\epsilon_1(n)$ | SSL helps |
|---|---|---|---|
| $\gamma \geq \gamma_0$ | $n^{-2\alpha/(2\alpha+d)}$ | $n^{-2\alpha/(2\alpha+d)}$ | No |
| $\gamma \geq c_o n^{-1/d}$ | $n^{-2\alpha/(2\alpha+d)}$ | $n^{-2\alpha/(2\alpha+d)}$ | No |
| $c_o n^{-1/d} > \gamma \geq C_o(\frac{m}{(\log m)^2})^{-1/d}$ | $n^{-2\alpha/(2\alpha+d)}$ | $n^{-1/d}$ | Yes |
| $C_o(\frac{m}{(\log m)^2})^{-1/d} > \gamma \geq -C_o(\frac{m}{(\log m)^2})^{-1/d}$ | $n^{-1/d}$ | $n^{-1/d}$ | No |
| $-C_o(\frac{m}{(\log m)^2})^{-1/d} > \gamma$ | $n^{-2\alpha/(2\alpha+d)}$ | $n^{-1/d}$ | Yes |
| $-\gamma_0 > \gamma$ | $n^{-2\alpha/(2\alpha+d)}$ | $n^{-1/d}$ | Yes |

If $\gamma$ is large relative to the average spacing between labeled data points $n^{-1/d}$, then a supervised learner can discern the decision sets accurately and SSL provides no gain. However, if $\gamma > 0$ is small relative to $n^{-1/d}$, but large with respect to the spacing between unlabeled data points $m^{-1/d}$, then the proposed semi-supervised learner provides improved error bounds compared to *any* supervised learner. If $|\gamma|$ is smaller than $m^{-1/d}$, the decision sets are not discernable with unlabeled data and SSL provides no gain. However, notice that the performance of the semi-supervised learner is no worse than the minimax lower bound for supervised learners. In the $\gamma < 0$ case, if $-\gamma$ larger than $m^{-1/d}$, then the semi-supervised learner can discern the decision sets and achieves smaller error bounds, whereas these sets cannot be as accurately discerned by any supervised learner. For the overlap case ($\gamma < 0$), supervised learners are always limited by the error incurred due to averaging across decision sets ($n^{-1/d}$). In particular, for the collection of distributions with $\gamma < -\gamma_0$, a faster rate of error convergence is attained by SSL compared to SL, provided $m \gg n^{2d}$.

## 6  Conclusions

In this paper, we develop a framework for evaluating the performance gains possible with semi-supervised learning under a cluster assumption using finite sample error bounds. The theoretical characterization we present explains why in certain situations unlabeled data can help to improve learning, while in other situations they may not. We demonstrate that there exist general situations under which semi-supervised learning can be significantly superior to supervised learning in terms of achieving smaller finite sample error bounds than any supervised learner, and sometimes in terms of a better rate of error convergence. Moreover, our results also provide a quantification of the relative value of unlabeled to labeled data. While we focus on the cluster assumption in this paper, we conjecture that similar techniques can be applied to quantify the performance of semi-supervised learning under the manifold assumption as well. In particular, we believe that the use of minimax lower bounding techniques is essential because many of the interesting distinctions between supervised and semi-supervised learning occur only in finite sample regimes, and rates of convergence and asymptotic analyses may not capture the complete picture.

## 7  Proofs

We sketch the main ideas behind the proofs here, please refer to [13] for details. Since the component densities are bounded from below and above, define $p_{\min} := b \min_k a_k \leq p(x) \leq B =: p_{\max}$.

### 7.1  Proof of Lemma 1

First, we state two relatively straightforward results about the proposed kernel density estimator.

**Theorem 1** (Sup-norm density estimation of non-boundary points). *Consider the kernel density estimator $\widehat{p}(x)$ proposed in Section 3. If the kernel $G$ satisfies supp$(G) = [-1,1]^d$, $0 < G \leq G_{\max} < \infty$, $\int_{[-1,1]^d} G(u)du = 1$ and $\int_{[-1,1]^d} u^j G(u)du = 0$ for $1 \leq j \leq [\alpha]$, then for all*

$p \in \mathcal{P}_X$, with probability at least $1 - 1/m$,

$$\sup_{x \in supp(p) \setminus \mathcal{B}} |p(x) - \widehat{p}(x)| = O\left(h_m^{\min(1,\alpha)} + \sqrt{\log m/(mh_m^d)}\right) =: \epsilon_m.$$

Notice that $\epsilon_m$ decreases with increasing $m$. A detailed proof can be found in [13].

**Corollary 2** (Empirical density of unlabeled data). *Under the conditions of Theorem 1, for all $p \in \mathcal{P}_X$ and large enough $m$, with probability $> 1 - 1/m$, for all $x \in supp(p) \setminus \mathcal{B}$, $\exists X_i \in \mathcal{U}$ s.t. $\|X_i - x\| \leq \sqrt{d}h_m$.*

*Proof.* From Theorem 1, for all $x \in supp(p) \setminus \mathcal{B}$, $\widehat{p}(x) \geq p(x) - \epsilon_m \geq p_{\min} - \epsilon_m > 0$, for $m$ sufficiently large. This implies $\sum_{i=1}^{m} G(H_m^{-1}(X_i - x)) > 0$, and $\exists X_i \in \mathcal{U}$ within $\sqrt{d}h_m$ of $x$. $\square$

Using the density estimation results, we now show that if $|\gamma| > 6\sqrt{d}h_m$, then for all $p \in \mathcal{P}_X$, all pairs of points $x_1, x_2 \in supp(p) \setminus \mathcal{B}$ and all $D \in \mathcal{D}$, for large enough $m$, with probability $> 1 - 1/m$, we have $x_1, x_2 \in D$ if and only if $x_1 \leftrightarrow x_2$. We establish this in two steps:

**1.** $x_1 \in D, x_2 \notin D \Rightarrow x_1 \nleftrightarrow x_2$ : Since $x_1$ and $x_2$ belong to different decision sets, all sequences connecting $x_1$ and $x_2$ through unlabeled data points pass through a region where either (i) the density is zero and since the region is at least $|\gamma| > 6\sqrt{d}h_m$ wide, there cannot exist a sequence as defined in Section 3 such that $\|z_j - z_{j+1}\| \leq 2\sqrt{d}h_m$, or (ii) the density is positive. In the latter case, the marginal density $p(x)$ jumps by at least $p_{\min}$ one or more times along all sequences connecting $x_1$ and $x_2$. Suppose the first jump occurs where decision set $D$ ends and another decision set $D' \neq D$ begins (in the sequence). Then since $D'$ is at least $|\gamma| > 6\sqrt{d}h_m$ wide, by Corollary 2 for all sequences connecting $x_1$ and $x_2$ through unlabeled data points, there exist points $z_i, z_j$ in the sequence that lie in $D \setminus \mathcal{B}$, $D' \setminus \mathcal{B}$, respectively, and $\|z_i - z_j\| \leq h_m \log m$. Since the density on each decision set is Hölder-$\alpha$ smooth, we have $|p(z_i) - p(z_j)| \geq p_{\min} - O((h_m \log m)^{\min(1,\alpha)})$. Since $z_i, z_j \notin \mathcal{B}$, using Theorem 1, $|\widehat{p}(z_i) - \widehat{p}(z_j)| \geq |p(z_i) - p(z_j)| - 2\epsilon_m > \delta_m$ for large enough $m$. Thus, $x_1 \nleftrightarrow x_2$.

**2.** $x_1, x_2 \in D \Rightarrow x_1 \leftrightarrow x_2$ : Since $D$ has width at least $|\gamma| > 6\sqrt{d}h_m$, there exists a region of width $> 2\sqrt{d}h_m$ contained in $D \setminus \mathcal{B}$, and Corollary 2 implies that with probability $> 1 - 1/m$, there exist sequence(s) contained in $D \setminus \mathcal{B}$ connecting $x_1$ and $x_2$ through $2\sqrt{d}h_m$-dense unlabeled data points. Since the sequence is contained in $D$ and the density on $D$ is Hölder-$\alpha$ smooth, we have for all points $z_i, z_j$ in the sequence that satisfy $\|z_i - z_j\| \leq h_m \log m$, $|p(z_i) - p(z_j)| \leq O((h_m \log m)^{\min(1,\alpha)})$. Since $z_i, z_j \notin \mathcal{B}$, using Theorem 1, $|\widehat{p}(z_i) - \widehat{p}(z_j)| \leq |p(z_i) - p(z_j)| + 2\epsilon_m \leq \delta_m$ for large enough $m$. Thus, $x_1 \leftrightarrow x_2$. $\square$

## 7.2 Proof of Corollary 1

Let $\Omega_1$ denote the event under which Lemma 1 holds. Then $P(\Omega_1^c) \leq 1/m$. Let $\Omega_2$ denote the event that the test point $X$ and training data $X_1, \ldots, X_n \in \mathcal{L}$ don't lie in $\mathcal{B}$. Then $P(\Omega_2^c) \leq (n+1)P(\mathcal{B}) \leq (n+1)p_{\max}\text{vol}(\mathcal{B}) = O(nh_m)$. The last step follows from the definition of the set $\mathcal{B}$ and since the boundaries of the support sets are Lipschitz, $K$ is finite, and hence $\text{vol}(\mathcal{B}) = O(h_m)$.

Now observe that $\widehat{f}_{\mathcal{D},n}$ essentially uses the clairvoyant knowledge of the decision sets $\mathcal{D}$ to discern which labeled points $X_1, \ldots, X_n$ are in the same decision set as $X$. Conditioning on $\Omega_1, \Omega_2$, Lemma 1 implies that $X, X_i \in D$ iff $X \leftrightarrow X_i$. Thus, we can define a semi-supervised learner $\widehat{f}_{m,n}$ to be the same as $\widehat{f}_{\mathcal{D},n}$ except that instead of using clairvoyant knowledge of whether $X, X_i \in D$, $\widehat{f}_{m,n}$ is based on whether $X \leftrightarrow X_i$. It follows that $\sup_{\mathcal{P}_{XY}(\gamma)} \mathbb{E}[\mathcal{E}(\widehat{f}_{m,n})|\Omega_1, \Omega_2] = \sup_{\mathcal{P}_{XY}(\gamma)} \mathbb{E}[\mathcal{E}(\widehat{f}_{\mathcal{D},n})]$, and since the excess risk is bounded: $\sup_{\mathcal{P}_{XY}(\gamma)} \mathbb{E}[\mathcal{E}(\widehat{f}_{m,n})] \leq \sup_{\mathcal{P}_{XY}(\gamma)} \mathbb{E}[\mathcal{E}(\widehat{f}_{m,n})|\Omega_1, \Omega_2] + O(1/m + nh_m)$. $\square$

## 7.3 Density adaptive Regression results

1) **Semi-Supervised Learning Upper Bound:** The clairvoyant counterpart of $\widehat{f}_{m,n}(x)$ is given as $\widehat{f}_{\mathcal{D},n}(x) \equiv \widehat{f}_{\mathcal{D},n,x}(x)$, where $\widehat{f}_{\mathcal{D},n,x}(\cdot) = \arg\min_{f' \in \Gamma} \sum_{i=1}^{n} (Y_i - f'(X_i))^2 \mathbf{1}_{x,X_i \in D} + \text{pen}(f')$, and is a standard supervised learner that performs piecewise polynomial fit on each decision set, where the regression function is Hölder-$\alpha$ smooth. Let $n_D = \frac{1}{n} \sum_{i=1}^{n} \mathbf{1}_{X_i \in D}$. It follows [12] that

$$\mathbb{E}[(f(X) - \widehat{f}_{\mathcal{D},n}(X))^2 \mathbf{1}_{X \in D} | n_D] \leq C \left(n_D/\log n_D\right)^{-\frac{2\alpha}{d+2\alpha}}.$$

Since $\mathbb{E}[(f(X) - \widehat{f}_{\mathcal{D},n}(X))^2] = \sum_{D \in \mathcal{D}} \mathbb{E}[(f(X) - \widehat{f}_{\mathcal{D},n}(X))^2 \mathbf{1}_{X \in D}] P(D)$, taking expectation over $n_D \sim$ Binomial$(n, P(D))$ and summing over all decision sets recalling that $|\mathcal{D}|$ is a finite constant, the overall error of $\widehat{f}_{\mathcal{D},n}$ scales as $n^{-2\alpha/(2\alpha+d)}$, ignoring logarithmic factors. If $|\gamma| > C_o(m/(\log m)^2)^{-1/d}$, using Corollary 1, the same performance bound holds for $\widehat{f}_{m,n}$ provided $m \gg n^{2d}$. See [13] for further details. If $|\gamma| < C_o(m/(\log m)^2)^{-1/d}$, the decision sets are not discernable using unlabeled data. Since the regression function is piecewise Hölder-$\alpha$ smooth on each empirical decision set, Using Theorem 9 in [12] and similar analysis, an upper bound of $\max\{n^{-2\alpha/(2\alpha+d)}, n^{-1/d}\}$ follows, which scales as $n^{-1/d}$ when $d \geq 2\alpha/(2\alpha - 1)$.

2) **Supervised Learning Lower Bound:** The formal minimax proof requires construction of a finite subset of distributions in $\mathcal{P}_{XY}(\gamma)$ that are the hardest cases to distinguish based on a finite number of labeled data $n$, and relies on a Hellinger version of Assouad's Lemma (Theorem 2.10 (iii) in [16]). Complete details are given in [13]. Here we present the simple intuition behind the minimax lower bound of $n^{-1/d}$ when $\gamma < c_o n^{-1/d}$. In this case the decision boundaries can only be localized to an accuracy of $n^{-1/d}$, the average spacing between labeled data points. Since the boundaries are Lipschitz, the expected volume that is incorrectly assigned to any decision set is $> c_1 n^{-1/d}$, where $c_1 > 0$ is a constant. Thus, if the expected excess risk at a point that is incorrectly assigned to a decision set can be greater than a constant $c_2 > 0$, then the overall expected excess risk is $> c_1 c_2 n^{-1/d}$. This is the case for both regression and binary classification. If $\gamma > c_o n^{-1/d}$, the decision sets can be accurately discerned from the labeled data alone. In this case, it follows that the minimax lower bound is equal to the minimax lower bound for Hölder-$\alpha$ smooth regression functions, which is $cn^{-2\alpha/(d+2\alpha)}$, where $c > 0$ is a constant [17].

## Footnotes

[1]This form is a slight generalization of the boundary fragment class of sets which is used as a common tool for analysis of learning problems [11]. Boundary fragment sets capture the salient characteristics of more general decision sets since, locally, the boundaries of general sets are like fragments in a certain orientation.

[2]If the component marginal densities and regression functions have different smoothnesses, say $\alpha$ and $\beta$, the same analysis holds except that $f(x)$ is Hölder-$\min(\alpha, \beta)$ smooth on each $D \in \mathcal{D}$.

## References

[1] Balcan, M.F., Blum, A.: A PAC-style model for learning from labeled and unlabeled data. In: 18th Annual Conference on Learning Theory, COLT. (2005)

[2] Kääriäinen, M.: Generalization error bounds using unlabeled data. In: 18th Annual Conference on Learning Theory, COLT. (2005)

[3] Rigollet, P.: Generalization error bounds in semi-supervised classification under the cluster assumption. Journal of Machine Learning Research **8** (2007) 1369–1392

[4] Lafferty, J., Wasserman, L.: Statistical analysis of semi-supervised regression. In: Advances in Neural Information Processing Systems 21, NIPS. (2007) 801–808

[5] Ben-David, S., Lu, T., Pal, D.: Does unlabeled data provably help? worst-case analysis of the sample complexity of semi-supervised learning. In: 21st Annual Conference on Learning Theory, COLT. (2008)

[6] Niyogi, P.: Manifold regularization and semi-supervised learning: Some theoretical analyses. Technical Report TR-2008-01, Computer Science Department, University of Chicago. URL http://people.cs.uchicago.edu/~niyogi/papersps/ssminimax2.pdf (2008)

[7] Seeger, M.: Learning with labeled and unlabeled data. Technical report, Institute for ANC, Edinburgh, UK. URL http://www.dai.ed.ac.uk/~seeger/papers.html (2000)

[8] Castelli, V., Cover, T.M.: On the exponential value of labeled samples. Pattern Recognition Letters **16**(1) (1995) 105–111

[9] Castelli, V., Cover, T.M.: The relative value of labeled and unlabeled samples in pattern recognition. IEEE Transactions on Information Theory **42**(6) (1996) 2102–2117

[10] Bickel, P.J., Li, B.: Local polynomial regression on unknown manifolds. In: IMS Lecture NotesMonograph Series, Complex Datasets and Inverse Problems: Tomography, Networks and Beyond. Volume 54. (2007) 177–186

[11] Korostelev, A.P., Tsybakov, A.B.: Minimax Theory of Image Reconstruction. Springer, NY (1993)

[12] Castro, R., Willett, R., Nowak, R.: Faster rates in regression via active learning. Technical Report ECE-05-03, ECE Department, University of Wisconsin - Madison. URL http://www.ece.wisc.edu/~nowak/ECE-05-03.pdf (2005)

[13] Singh, A., Nowak, R., Zhu, X.: Finite sample analysis of semi-supervised learning. Technical Report ECE-08-03, ECE Department, University of Wisconsin - Madison. URL http://www.ece.wisc.edu/~nowak/SSL_TR.pdf (2008)

[14] Korostelev, A., Nussbaum, M.: The asymptotic minimax constant for sup-norm loss in nonparametric density estimation. Bernoulli **5(6)** (1999) 1099–1118

[15] Chapelle, O., Zien, A.: Semi-supervised classification by low density separation. In: Tenth International Workshop on Artificial Intelligence and Statistics. (2005) 57–64

[16] Tsybakov, A.B.: Introduction a l'estimation non-parametrique. Springer, Berlin Heidelberg (2004)

[17] Stone, C.J.: Optimal rates of convergence for nonparametric estimators. The Annals of Statistics **8(6)** (1980) 1348–1360

